# MAS: a multiplicative approximation scheme for probabilistic inference

**Ydo Wexler**
Microsoft Research
Redmond, WA 98052
ydow@microsoft.com

**Christopher Meek**
Microsoft Research
Redmond, WA 98052
meek@microsoft.com

## Abstract

We propose a multiplicative approximation scheme (MAS) for inference problems in graphical models, which can be applied to various inference algorithms. The method uses $\epsilon$-decompositions which decompose functions used throughout the inference procedure into functions over smaller sets of variables with a known error $\epsilon$. MAS translates these local approximations into bounds on the accuracy of the results. We show how to optimize $\epsilon$-decompositions and provide a fast closed-form solution for an $L_2$ approximation. Applying MAS to the Variable Elimination inference algorithm, we introduce an algorithm we call DynaDecomp which is extremely fast in practice and provides guaranteed error bounds on the result. The superior accuracy and efficiency of DynaDecomp is demonstrated.

## 1   Introduction

Probabilistic graphical models gained popularity in the recent decades due to their intuitive representation and because they enable the user to query about the value distribution of variables of interest [19]. Although very appealing, these models suffer from the problem that performing inference in the model (e.g. computing marginal probabilities or its likelihood) is NP-hard [6].

As a result, a variety of approximate inference methods have been developed. Among these methods are loopy message propagation algorithms [24], variational methods [16, 12], mini buckets [10], edge deletion [8], and a variety of Monte Carlo sampling techniques  [13, 19, 21, 4, 25]. Approximation algorithms that have useful error bounds and speedup while maintaining high accuracy, include the work of Dechter and colleagues [2, 3, 10, 17], which provide both upper and lower bounds on probabilities, upper bounds suggested by Wainwright et.al. [23], and variational lower bounds [16].

In this paper we present an approximation scheme called the Multiplicative Approximation Scheme (MAS), that provides error bounds for the computation of likelihood of evidence, marginal probabilities, and the Maximum Probability Explanation (MPE) in discrete directed and undirected graphical models. The approximation is based on a local operation called an $\epsilon$-decomposition, that decomposes functions used in the inference procedure into functions over smaller subsets of variables, with a guarantee on the error introduced. The main difference from existing approximations is the ability to translate the error introduced in the local decompositions performed during execution of the algorithm into bounds on the accuracy of the entire inference procedure. We note that this approximation can be also applied to the more general class of multiplicative models introduced in [27].

We explore optimization of $\epsilon$-decompositions and provide a fast optimal closed form solution for the $L_2$ norm. We also show that for the Kullback-Leiber divergence the optimization problem can be solved using variational algorithms on local factors. MAS can be applied to various inference algorithms. As an example we show how to apply MAS to the Variable Elimination (VE) algorithm [9, 20], and present an algorithm called DynaDecomp, which dynamically decomposes functions in the VE algorithm. In the results section we compare the performance of DynaDecomp with that of Mini-buckets [10], GMF [28] and variational methods [26] for various types of models. We find that our method achieves orders of magnitude better accuracy on all datasets.

## 2   Multiplicative Approximation Scheme (MAS)

We propose an approximation scheme, called the Multiplicative Approximation Scheme (MAS) for inference problems in graphical models. The basic operations of the scheme are local approximations called $\epsilon$-decompositions that decouple the dependency of variables. Every such local decomposition has an associated error that our scheme combines into an error bound on the result.

Consider a graphical model for $n$ variable $X = \{X_1, \ldots, X_n\}$ that encodes a probability distribution $P(X) = \prod_j \psi_j(d_j)$ where $D_j \subseteq X$ are sets determined by the model. Throughout the paper we denote variables and sets of variables with capital letters and denote a value assigned to them with lowercase letters. We denote the observed variables in the model by $E = X \setminus H$ where $E = e$. To simplify the proofs we assume $\psi_j(d_j) > 1$. When this is not the case, as in BNs, every function $\psi_j$ can be multiplied by a constant $z_j$ such that the assumption holds, and the result is obtained after dividing by $\prod_j z_j$. Thus, here we assume positivity but discuss how this can be relaxed below.

In addition to approximating functions $\psi$ by which the original model is defined, we also may wish to approximate other functions such as intermediate functions created in the course of an inference algorithm. We can write the result of marginalizing out a set of hidden variables as a factor of functions $f_i$. The log of the probability distribution the model encodes after such marginalization can then be written as

$$\log P(A, E) = \log \prod_i f_i(U_i) = \sum_i \phi_i(U_i) \tag{1}$$

where $A \subseteq H$. When $A = H$ we can choose sets $U_i = D_i$ and functions $f_i(U_i) = \psi_i(D_i)$.

**Definition 1 ($\epsilon$-decomposition)**  *Given a set of variables $W$, and a function $\phi(W)$ that assigns real values to every instantiation $W = w$, a set of $m$ functions $\tilde{\phi}_l(W_l)$, $l = 1 \ldots m$, where $W_l \subseteq W$ is an $\epsilon$-decomposition if $\bigcup_l W_l = W$, and*

$$\frac{1}{1+\epsilon} \le \frac{\sum_l \tilde{\phi}_l(w_l)}{\phi(w)} \le 1 + \epsilon \tag{2}$$

*for some $\epsilon \ge 0$, where $w_l$ is the projection of $w$ on $W_l$.*

Note that an $\epsilon$-decomposition is not well defined for functions $\phi$ that equal zero or are infinite for some instantiations. These functions can still be $\epsilon$-decomposed for certain choices of subsets $W_l$ by defining $\frac{0}{0} = 1$ and $\frac{\infty}{\infty} = 1$. We direct the interested reader to the paper of Geiger et.al. [12] for a discussion on choosing such subsets. We also note that when approximating models in which some assignments have zero probability, the theoretical error bounds can be arbitrarily bad, yet, in practice the approximation can sometimes yield good results.

The following theorems show that using $\epsilon$-decompositions the log-likelihood, $\log P(e)$, log of marginal probabilities, the log of the Most Probable Explanation (MPE) and the log of the Maximum Aposteriori Probability (MAP) can all be approximated within a multiplicative factor using a set of $\epsilon$-decompositions.

**Lemma 1**  *Let $A \subseteq H$, and let $P(A, E)$ factor according to Eq. 1, then the log of the joint probability $P(a, e)$ can be approximated within a multiplicative factor of $1 + \epsilon_{max}$ using a set of $\epsilon_i$-decompositions, where $\epsilon_{max} = \max_i\{\epsilon_i\}$.*

*Proof:*

$$\log \tilde{P}(a, e) \equiv \log \prod_{i,l} e^{\tilde{\phi}_{il}(u_{il})} = \sum_{i,l} \tilde{\phi}_{il}(u_{il}) \le \sum_i (1 + \epsilon_i)\phi_i(u_i) \le (1 + \epsilon_{max})\log P(a, e)$$

$$\log \tilde{P}(a, e) \equiv \log \prod_{i,l} e^{\tilde{\phi}_{il}(u_{il})} = \sum_{i,l} \tilde{\phi}_{il}(u_{il}) \ge \sum_i \frac{1}{1+\epsilon_i}\phi_i(u_i) \ge \frac{1}{1+\epsilon_{max}}\log P(a, e) \quad \blacksquare$$

**Theorem 1**  *For a set $A' \subseteq A$ the expression $\log \sum_{a'} P(a, e)$ can be approximated within a multiplicative factor of $1 + \epsilon_{max}$ using a set of $\epsilon_i$-decompositions.*

*Proof:* Recall that $\sum_j (c_j)^r \leq \left( \sum_j c_j \right)^r$ for any set of numbers $c_j \geq 0$ and $r \geq 1$. Therefore, using Lemma 1 summing out any set of variables $A' \subseteq A$ does not increase the error:

$$\log \sum_{a'} \tilde{P}(a,e) \leq \log \sum_{a'} \left( \prod_i e^{\phi_i(u_i)} \right)^{1+\epsilon_{max}} \leq \log \left( \sum_{a'} \prod_i e^{\phi_i(u_i)} \right)^{1+\epsilon_{max}} = (1+\epsilon_{max}) \log \sum_{a'} P(a,e)$$

Similarly for the upper bound approximation we use the fact that $\sum_j (c_j)^r \geq \left( \sum_j c_j \right)^r$ for any set of numbers $c_j \geq 0$ and $0 < r \leq 1$. ■

Note that whenever $E = \emptyset$, Theorem 1 claims that the log of all marginal probabilities can be approximated within a multiplicative factor of $1 + \epsilon_{max}$. In addition, for any $E \subseteq X$ by setting $A' = A$ the log-likelihood $\log P(e)$ can be approximated with the same factor.

A similar analysis can also be applied with minor modifications to the computation of related problems like the MPE and MAP. We adopt the simplification of the problems suggested in [10], reducing the problem of the Most Probable Explanation (MPE) to computing $P(h^*, e) = \max_h P(h, e)$ and the problem of the Maximum Aposteriori Probability (MAP) to computing $P(a^*, e) = \max_a \sum_{H \backslash A = h^-} P(h, e)$ for a set $A \subseteq H$.

Denote the operator $\oplus$ as either a sum or a $\max$ operator. Then, similar to Eq. 1, for a set $H' \subseteq H$ we can write

$$\log \oplus_{h'} P(h, e) = \log \prod_i f_i(U_i) = \sum_i \phi_i(U_i) \tag{3}$$

**Theorem 2** *Given a set $A \subseteq H$, the log of the MAP probability $\log \max_a \sum_{H \backslash A = h^-} P(h, e)$ can be approximated within a multiplicative factor of $1 + \epsilon_{max}$ using a set of $\epsilon_i$-decompositions.*

*Proof:* The proof follows that of Theorem 1 with the addition of the fact that $\max_j (c_j)^r = (\max_j c_j)^r$ for any set of real numbers $c_j \geq 0$ and $r \geq 0$. ■

An immediate conclusion from Theorem 2 is that the MPE probability can also be approximated with the same error bounds, by choosing $A = H$.

## 2.1 Compounded Approximation

The results on using $\epsilon$-decompositions assume that we decompose functions $f_i$ as in Eqs. 1 and 3. Here we consider decompositions of any function created during the inference procedure, and in particular compounded decompositions of functions that were already decomposed. Suppose that a function $\tilde{\phi}(W)$, that already incurs an error $\epsilon_1$ compared to a function $\phi(W)$, can be decomposed with an error $\epsilon_2$. Then, according to Eq. 2, this results in a set of functions $\hat{\phi}_l(W_l)$, such that the error of $\sum_l \hat{\phi}_l(W_l)$ is $(1 + \epsilon_1) \cdot (1 + \epsilon_2)$ wrt $\phi(W)$.

To understand what is the guaranteed error for an entire inference procedure consider a directed graph where the nodes represent functions of the inference procedure, and each node $v$ has an associated error $r_v$. The nodes representing the initial potential functions of the model $\psi_i$ have no parents in the model and are associated with zero error ($r_v = 1$). Every multiplication operation is denoted by edges directed from the nodes $S$, representing the multiplied functions, to a node $t$ representing the resulting function, the error of which is $r_t = \max_{s \in S} r_s$. An $\epsilon$-decomposition on the other hand has a single source node $s$ with an associated error $r_s$, representing the decomposed function, and several target nodes $T$, with an error $r_t = (1 + \epsilon) r_s$ for every $t \in T$. The guaranteed error for the entire inference procedure is then the error associated with the sink function in the graph. In Figure 1 we illustrate such a graph for an inference procedure that starts with four functions ($f_a$, $f_b$, $f_c$ and $f_d$) and decomposes three functions, $f_a, f_g$ and $f_j$, with errors $\epsilon_1, \epsilon_2$ and $\epsilon_3$ respectively. In this example we assume that $\epsilon_1 > \epsilon_2$ and that $1 + \epsilon_1 < (1 + \epsilon_2)(1 + \epsilon_3)$.

## 2.2 $\epsilon$-decomposition Optimization

$\epsilon$-decompositions can be utilized in inference algorithms to reduce the computational cost by parsimoniously approximating factors that occur during the course of computation. As we discuss in

Section 3, both the selection of the form of the $\epsilon$-decomposition (i.e., the sets $W_i$) and which factors to approximate impact the overall accuracy and runtime of the algorithm. Here we consider the problem of optimizing the approximating functions $\tilde{\phi}_i$ given a selected factorization $W_i$.

Given a function $f(W) = e^{\phi(W)}$ and the sets $W_i$, the goal is to optimize the functions $\phi_i(W_i)$ in order to minimize the error $\epsilon_f$ introduced in the decomposition. The objective function is therefore

$$\min_{(\tilde{\phi}_1,\ldots,\tilde{\phi}_m)} \max_{w \in W} \left\{ \frac{\sum_i \tilde{\phi}_i(w_i)}{\phi(w)}, \frac{\phi(w)}{\sum_i \tilde{\phi}_i(w_i)} \right\} \tag{4}$$

This problem can be formalized as a convex problem using the following notations.

Let $t = \max_{w \in W} \left\{ \frac{\sum_i \tilde{\phi}_i(w_i)}{\phi(w)}, \frac{\phi(w)}{\sum_i \tilde{\phi}_i(w_i)} \right\}$ and $S_w = \frac{\phi(w)}{\sum_i \tilde{\phi}_i(w_i)}$. Now we can reformulate the problem as

$$\min_{(\tilde{\phi}_1,\ldots,\tilde{\phi}_m)} t \quad \text{s.t. } \forall (W = w) \quad S_w \leq t \quad \text{and} \quad S_w^{-1} \leq t \tag{5}$$

This type of problems can be solved with geometric programming techniques, and in particular using interior-point methods [18]. Unfortunately, in the general case the complexity of solving this problem requires $O(m^3|W|^3)$ time, and hence can be too expensive for functions over a large domain. On the other hand, many times functions defined over a small domain can not be decomposed without introducing a large error. Thus, when trying to limit the error introduced, a significant amount of time is needed for such optimization. To reduce the computational cost of the optimization we resort to minimizing similar measures, in the hope that they will lead to a small error $\epsilon_f$.

Note that by deviating from Eq. 4 to choose the functions $\tilde{\phi}_i$ we may increase the worst case penalty error but not necessarily the actual error achieved by the approximation. In addition, even when using different measures for the optimization we can still compute $\epsilon_f$ exactly.

### 2.2.1 Minimizing the $L_2$ Norm

An alternative minimization measure, the $L_2$ norm, is closely related to that in Eq. 4 and given as:

$$\min_{(\tilde{\phi}_1,\ldots,\tilde{\phi}_m)} \sqrt{ \sum_{w \in W} \left[ \left( \sum_i \tilde{\phi}_i(w_i) \right) - \phi(w) \right]^2 } \tag{6}$$

We give a closed form analytic solution for this minimization problem when the sets $W_i$ are disjoint, but first we can remove the square root from the optimization formula due to the monotonicity of the square root for positive values. Hence we are left with the task of minimizing:

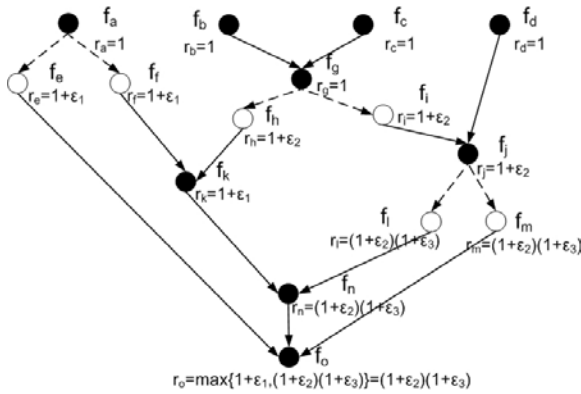

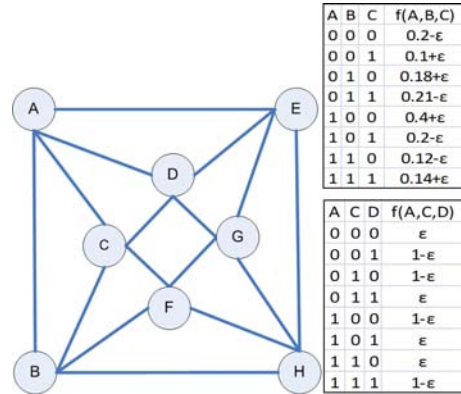

| A | B | C | f(A,B,C) |
|---|---|---|---|
| 0 | 0 | 0 | 0.2-ε |
| 0 | 0 | 1 | 0.1+ε |
| 0 | 1 | 0 | 0.18+ε |
| 0 | 1 | 1 | 0.21-ε |
| 1 | 0 | 0 | 0.4+ε |
| 1 | 0 | 1 | 0.2-ε |
| 1 | 1 | 0 | 0.12-ε |
| 1 | 1 | 1 | 0.14+ε |

| A | C | D | f(A,C,D) |
|---|---|---|---|
| 0 | 0 | 0 | ε |
| 0 | 0 | 1 | 1-ε |
| 0 | 1 | 0 | 1-ε |
| 0 | 1 | 1 | ε |
| 1 | 0 | 0 | 1-ε |
| 1 | 0 | 1 | ε |
| 1 | 1 | 0 | ε |
| 1 | 1 | 1 | 1-ε |

Figure 1: A schematic description of an inference procedure along with the associated error. The procedure starts with four functions ($f_a, f_b, f_c$ and $f_d$) and decomposes three functions, $f_a, f_g$ and $f_j$, with errors $\epsilon_1, \epsilon_2$ and $\epsilon_3$ respectively. In this example we assume that $\epsilon_1 > \epsilon_2$, which results in an error $r_k = 1 + \epsilon_1$, and assume that $1 + \epsilon_1 < (1 + \epsilon_2)(1 + \epsilon_3)$, which results in the errors $r_m = r_o = (1 + \epsilon_2)(1 + \epsilon_3)$.

Figure 2: An irreducible minor graph of a $4 \times 4$ Ising model that can be obtained via VE without creating functions of more than 3 variables. Applying MAS, only one function over three variables needs to be decomposed into two functions over overlapping sets of variables in order to complete inference using only functions over three or less variables.

$$\min_{(\tilde{\phi}_1,...,\tilde{\phi}_m)} \sum_{w \in W} \left[ \left( \sum_i \tilde{\phi}_i(w_i) \right) - \phi(w) \right]^2 \qquad (7)$$

We use the notation $w \approx w_k$ to denote an instantiation $W = w$ that is consistent with the instantiation $W_k = w_k$. To find the optimal value of $\tilde{\phi}_i(w_i)$ we differentiate Eq. 7 with respect to each $\tilde{\phi}_k(w_k)$ and set to zero. Choosing the constraint $\sum_w \tilde{\phi}_i(w_i) = \frac{\sum_w \phi(w)}{m}$ in the resulting under-constrained set of linear equations we get

$$\tilde{\phi}_k(w_k) = \frac{\sum\limits_{w \approx w_k} \phi(w)}{\prod\limits_{i \neq k} |W_i|} - \sum_{i \neq k} \frac{\sum\limits_w \phi(w)}{m \prod\limits_j |W_j|}$$

As the last term is independent of the index $i$ we finally obtain

$$\tilde{\phi}_k(w_k) = \frac{\sum\limits_{w \approx w_k} \phi(w)}{\prod\limits_{i \neq k} |W_i|} - \frac{(m-1) \sum\limits_w \phi(w)}{m|W|} \qquad (8)$$

The second term of Eq. 8 is computed once for a decomposition operation. Denoting $|W| = N$ this term can be computed in $O(N)$ time. Computing the first term of Eq. 8 also takes $O(N)$ time but it needs to be computed for every resulting function $\tilde{\phi}_k$, hence taking an overall time of $O(Nm)$.

### 2.2.2 Minimizing the KL Divergence

The Kulback-liebert (KL) divergence is another common alternative measure used for optimization:

$$\min_{(\tilde{\phi}_1,...,\tilde{\phi}_m)} \sum_{w \in W} \left[ \sum_i \tilde{\phi}_i(w_i) \right] \log \frac{\sum_i \tilde{\phi}_i(w_i)}{\phi(w)} \qquad (9)$$

Although no closed form solution is known for this minimization problem, iterative algorithms were devised for variational approximation, which start with arbitrary functions $\tilde{\phi}_i(W_i)$ and converge to a local minimum [16, 12]. Despite the drawbacks of unbounded convergence time and lack of guarantee to converge to the global optimum, these methods have proven quite successful. In our context this approach has the benefit of allowing overlapping sets $W_i$.

## 3 Applying MAS to Inference Algorithms

Our multiplicative approximation scheme offers a way to reduce the computational cost of inference by decoupling variables via $\epsilon$-decompositions. The fact that many existing inference algorithms compute and utilize multiplicative factors during the course of computation means that the scheme can be applied widely. The approach does require a mechanism to select functions to decompose, however, the flexibility of the scheme allows a variety of alternative mechanisms. One simple cost-focused strategy is to decompose a function whenever its size exceeds some threshold. An alternative quality-focused strategy is to choose an $\epsilon$ and search for $\epsilon$-decompositions $W_i$. Below we consider the application of our approximation scheme to variable elimination with yet another selection strategy. We note that heuristics for choosing approximate factorizations exist for the selection of disjoint sets [28] and for overlapping sets [5] and could be utilized. The ideal application of our scheme is likely to depend both on the specific inference algorithm and the application of interest.

### 3.1 Dynamic Decompositions

One family of decomposition strategies which are of particular interest, are those which allow for dynamic decompositions during the inference procedure. In this dynamic framework, MAS can be incorporated into known exact inference algorithms for graphical models, provided that local functions can be bounded according to Eq. 2. A dynamic decomposition strategy applies $\epsilon$-decompositions to functions in which the original model is defined and to intermediate functions created in the course of the inference algorithm, according to Eq. 1 or Eq. 3, based on the current state of the algorithm, and the accuracy introduced by the possible decompositions.

Unlike other approximation methods, such as the variational approach [16] or the edge deletion approach [8], dynamic decompositions has the capability of decoupling two variables in some contexts while maintaining their dependence in others. If we wish to restrict ourselves to functions over three or less variables when performing inference on a $4 \times 4$ Ising model, the model in Figure 2 is an inevitable minor, and from this point of the elimination, approximation is mandatory. In the variational framework, an edge in the graph should be removed, disconnecting the direct dependence between two or more variables (e.g. removing the edge A-C would result in breaking the set ABC into the sets AB and BC and breaking the set ACD into AD and CD). The same is true for the edge deletion method, with the difference in the new potentials associated with the new sets. Dynamic decompositions allow for a more refined decoupling, where the dependence is removed only in some of the functions. In our example breaking the set ABC into AB and BC while keeping the set ACD intact is possible and is also sufficient for reducing the complexity of inference to functions of no more than three variables (the elimination order would be: A,B,F,H,C,E,D,G). Moreover, if decomposing the set ABC can be done with an error $\epsilon_{ABC}$, as defined in Eq. 2, then we are guaranteed not to exceed this error for the entire approximate inference procedure. An extreme example will be the functions for the sets ABC and ACD as appear in the tables of Figure 2. It is possible to decompose the function over the set ABC into two functions over the sets AB and BC with an arbitrarily small error, while the same is not possible for the function over the set ACD. Hence, in this example the result of our method will be nearly equal to the solution of exact inference on the model, and the theoretical error bounds will be arbitrarily small, while other approaches, such as the variational method, can yield arbitrarily bad approximations.

We discuss how to incorporate MAS into the Variable Elimination (VE) algorithm for computing the likelihood of a graphical model [9, 20]. In this algorithm variables $V \in H$ are summed out iteratively after multiplying all existing functions that include $V$, yielding intermediate functions $f(W \subseteq X)$ where $V \notin W$. MAS can be incorporated into the VE algorithm by identifying $\epsilon$-decompositions for some of the intermediate functions $f$. This results in the elimination of $f$ from the pool of functions and adding instead the functions $\tilde{f}_i(W_i) = e^{\tilde{\phi}_i(W_i)}$. Note that the sets $W_i$ are not necessarily disjoint and can have common variables. Using $\epsilon$-decompositions reduces the computational complexity, as some variables are decoupled in specific points during execution of the algorithm. Throughout the algorithm the maximal error $\epsilon_{max}$ introduced by the decompositions

Table 1: Accuracy and speedup for grid-like models. Upper panel: attractive Ising models; Middle panel: repulsive Ising models; Lower panel: Bayesian network grids with random probabilities.

| Model | Num Values | Accuracy | Bounds | Speedup | DD time (secs) |
|---|---|---|---|---|---|
| $10 \times 10$ | 5 | 2.4e-4 | 0.0096 | 49.2 | 0.04 |
| $10 \times 10$ | 2 | 2.1e-4 | 0.0094 | 2.5 | 0.01 |
| $15 \times 15$ | 5 | 1.2e-4 | 0.0099 | 223.3 | 0.21 |
| $15 \times 15$ | 2 | 2.2e-4 | 0.0096 | 8.3 | 0.04 |
| $20 \times 20$ | 2 | 1.2e-4 | 0.0095 | 12.9 | 0.08 |
| $25 \times 25$ | 2 | 2.6e-5 | 0.0092 | 20.9 | 0.10 |
| $30 \times 30$ | 2 | 5.7e-4 | 0.0097 | 236.7 | 0.11 |
| $10 \times 10$ | 5 | 3.2e-4 | 0.0099 | 38.2 | 0.04 |
| $10 \times 10$ | 5 | 3.5e-4 | 0.0098 | 2.3 | 0.01 |
| $15 \times 15$ | 5 | 3.2e-3 | 0.0099 | 568.4 | 0.12 |
| $15 \times 15$ | 2 | 8.6e-4 | 0.0094 | 7.2 | 0.05 |
| $20 \times 20$ | 2 | 4.5e-4 | 0.0091 | 14.3 | 0.10 |
| $25 \times 25$ | 2 | 3.1e-5 | 0.0094 | 22.8 | 0.11 |
| $30 \times 30$ | 2 | 8.1e-5 | 0.0099 | 218.7 | 0.10 |
| $10 \times 10$ | 2 | 3.0e-3 | 0.0098 | 1.1 | 0.01 |
| $12 \times 12$ | 2 | 8.1e-3 | 0.0096 | 11.3 | 0.02 |
| $15 \times 15$ | 2 | 1.7e-3 | 0.0098 | 201.4 | 0.05 |
| $18 \times 18$ | 2 | 3.0e-4 | 0.0090 | 1782.8 | 0.15 |
| $20 \times 20$ | 2 | 1.8e-3 | 0.0097 | 7112.9 | 1.30 |
| $10 \times 10$ | 5 | 2.8e-5 | 0.0095 | 49.3 | 0.03 |
| $12 \times 12$ | 5 | 5.5e-4 | 0.0096 | 458.6 | 0.05 |
| $7 \times 7$ | 10 | 1.8e-4 | 0.0093 | 7.8 | 0.03 |
| $8 \times 8$ | 10 | 1.4e-4 | 0.0098 | 8.4 | 0.15 |

**Algorithm 1**: DynaDecomp

**Input**: A model for $n$ variables $X = \{X_1, \ldots, X_n\}$ and functions $\psi_i(D_i \subseteq X)$, that encodes $P(X) = \prod_i \psi_i(D_i)$; A set $E = X \setminus H$ of observed variables and their assignment $E = e$; An elimination order $R$ over the variables in $H$; scalars $M$ and $\eta$.

**Output**: The log-likelihood $\log P(e)$; an error $\epsilon$.

Initialize: $\epsilon = 0$; $F \leftarrow \{\psi_i(D_i)\}$; $I(\psi_i) = false$;
**for** $i = 1$ *to* $n$ **do**
    $k \leftarrow R[i]$;
    $T \leftarrow \{f : f \text{ contains } X_k, f \in F\}$;
    $F \leftarrow F \setminus T$;
    $f' \leftarrow \sum_{x_k} \otimes(T)$;
    $I(f') = \bigwedge_{f \in T} I(f)$;
    **if** $|f'| \geq M$ *and* $I(f') = true$ **then**
        $(\epsilon_{f'}, \tilde{F}) \leftarrow \oslash(f')$;
        **if** $\epsilon_{f'} \leq \eta$ **then**
            $\forall \tilde{f} \in \tilde{F} \;\; I(\tilde{f}) = false$;
            $F \leftarrow F \cup \tilde{F}$;
            $\epsilon = \max\{\epsilon, \epsilon_{f'}\}$;
        **else**
            $F \leftarrow F \cup f'$;
    **else**
        $F \leftarrow F \cup f'$;
multiply all constant functions in $F$ and put in $p$;
return $\log p, \epsilon$;

can be easily computed by associating functions with errors, as explained in Section 2.1. In our experiments we restrict attention to non-compounded decompositions. Our algorithm decomposes a function only if it is over a given size $M$, and if it introduces no more than $\eta$ error. The approximating functions in this algorithm are strictly disjoint, of size no more than $\sqrt{M}$, and with the variables assigned randomly to the functions. We call this algorithm DynaDecomp (DD) and provide a pseudo-code in Algorithm 1. There we use the notation $\otimes(T)$ to denote multiplication of the functions $f \in T$, and $\oslash(f)$ to denote decomposition of function $f$. The outcome of $\oslash(f)$ is a pair $(\epsilon, \tilde{F})$ where the functions $\tilde{f}_i \in \tilde{F}$ are over a disjoint set of variables.

We note that MAS can also be used on top of other common algorithms for exact inference in probabilistic models which are widely used, thus gaining similar benefits as those algorithms. For example, applying MAS to the junction tree algorithm [14] a decomposition can decouple variables in messages sent from one node in the junction tree to another, and approximate all marginal distributions of single variables in the model in a single run, with similar guarantees on the error. This extension is analogous to how the mini-clusters algorithm [17] extends the mini-bucket algorithm [10].

## 4   Results

We demonstrate the power of MAS by reporting the accuracy and theoretical bounds for our DynaDecomp algorithm for a variety of models. Our empirical study focuses on approximating the likelihood of evidence, except when comparing to the results of Xing et. al. [28] on grid models. The quality of approximation is measured in terms of accuracy and speedup. The accuracy is reported as $\max\{\frac{\log L}{\log \tilde{L}}, \frac{\log \tilde{L}}{\log L}\} - 1$ where $L$ is the likelihood and $\tilde{L}$ is the approximate likelihood achieved by DynaDecomp. We also report the theoretical accuracy which is the maximum error introduced by decomposition operations. The speedup is reported as a ratio of run-times for obtaining the approximated and exact solutions, in addition to the absolute time of approximation. In all experiments a random partition was used to decompose the functions, and the $L_2$ norm optimization introduced in Section 2.2.1 was applied to minimize the error. The parameter $M$ was set to $10,000$ and the guaranteed accuracy $\eta$ was set to $1\%$, however, as is evident from the results, the algorithm usually achieves better accuracy.

We compared the performance of DynaDecomp with the any-time Mini-buckets (MB) algorithm [10]. The parameters $i$ and $m$, which are the maximal number of variables and functions in a mini-bucket, were initially set to $3$ and $1$ respectively. The parameter $\epsilon$ was set to zero, not constraining the possible accuracy. Generally we allowed MB to run the same time it took DynaDecomp to approximate the model, but not less than one iteration (with the initial parameters).

We used two types of grid-like models. The first is an Ising model with random attractive or repulsive pair-wise potentials, as was used in [28]. When computing likelihood in these models we randomly assigned values to $10\%$ of the variables in the model. The other kind of grids were Bayesian networks where every variable $X_{ij}$ at position $(i, j)$ in the grid has the variables $X_{i-1,j}$ and $X_{i,j-1}$ as parents in the model. In addition, every variable $X_{ij}$ has a corresponding observed variable $Y_{ij}$ connected to it. Probabilities in these models were uniformly distributed between zero and one. Inference on these models, often used in computer vision [11], is usually harder than on Ising models, due to reduced factorization. We used models where the variables had either two, five or ten values. The results are shown in Table 1. In addition, we applied DynaDecomp to two $100 \times 100$ Ising grid models with binary variables. Inference in these models is intractable. We estimate the time for exact computation using VE on current hardware to be $3 \cdot 10^{15}$ seconds. This is longer than the time since the disappearance of the dinosaurs. Setting $\eta$ to $2\%$, DynaDecomp computed the approximated likelihood in 7.09 seconds for the attractive model and 8.14 seconds for the repulsive one.

Comparing our results with those obtained by the MB algorithm with an equivalent amount of computations, we find that on the average the accuracy of MB across all models in Tables 1 is 0.198 while the average accuracy of DynaDecomp is $9.8e^{-4}$, more than 200 times better than that of MB. In addition the theoretical guarantees are more than $30\%$ for MB and $0.96\%$ for DynaDecomp, a 30-fold improvement. As a side note, the MB algorithm performed significantly better on attractive Ising models than on repulsive ones. To compare our results with those reported in [28] we computed all the marginal probabilities (without evidence) and calculated the $L_1$-based measure

$\sum_{i,j} \sum_{x_{ij}} P(x_{ij}) - \tilde{P}(x_{ij})$. Running on the Ising models DynaDecomp obtained an average of $1.86e^{-5}$ compared to $0.003$ of generalized belief propagation (GBP) and $0.366$ of generalized mean field (GMF). Although the run times are not directly comparable due to differences in hardware, DynaDecomp average run-time was less than $0.1$ seconds, while the run-time of GBP and GMF was previously reported [28] to be $140$ and $1.6$ seconds respectively, on $8 \times 8$ grids.

We applied our method to probabilistic phylogenetic models. Inference on these large models, which can contain tens of thousands of variables, is used for model selection purposes. Previous works [15, 26] have obtained upper and lower bounds on the likelihood of evidence in the models suggested in [22] using variational methods, reporting an error of $1\%$. Using the data as in [26], we achieved less than $0.01\%$ error on average within a few seconds, which improves over previous results by two orders of magnitude both in terms of accuracy and speedup.

In addition, we applied DynaDecomp to 24 models from the UAI'06 evaluation of probabilistic inference repository [1] with $\eta = 1\%$. Only models that did not have zeros and that our exact inference algorithm could solve in less than an hour were used. The average accuracy of DynaDecomp on these models was $0.0038$ with an average speedup of $368.8$ and average run-time of $0.79$ seconds. We also applied our algorithm to two models from the CPCS benchmark (cpcs360b and cpcs422b). DynaDecomp obtained an average accuracy of $0.008$ versus $0.056$ obtained by MB. We note that the results obtained by MB are consistent with those reported in [10] for the MPE problem.

## References

[1] Evaluation of probabilistic inference systems: http://tinyurl.com/3k9l4b, 2006.

[2] Bidyuk and Dechter. An anytime scheme for bounding posterior beliefs. *AAAI* 2006.

[3] Bidyuk and Dechter. Improving bound propagation. In *ECAI* 342–346, 2006.

[4] Cheng and Druzdzel. AIS-BN: An adaptive importance sampling algorithm for evidential reasoning in large Bayesian networks. *JAIR* 13:155–188, 2000.

[5] Choi and Darwiche. A variational approach for approximating Bayesian networks by edge deletion. *UAI* 2006.

[6] Cooper. The computational complexity of probabilistic inference using Bayesian belief networks. *AI* 42(2-3):393–405, 1990.

[7] Dagum and Luby. Approximating probabilistic inference in Bayesian belief networks is NP-hard. *AI*, 60(1):141–153, 1993.

[8] Darwiche, Chan, and Choi. On Bayesian network approximation by edge deletion. *UAI* 2005.

[9] Dechter. Bucket elimination: A unifying framework for reasoning. *AI* 113(1-2):41–85, 1999.

[10] Dechter and Rish. Mini-buckets:A general scheme for bounded inference. *J.ACM* 50:107–153, 2003.

[11] W. Freeman, W. Pasztor, and O. Carmichael. Learning low-level vision. *IJCV* 40:25–47, 2000.

[12] Geiger, Meek, and Wexler. A variational inference procedure allowing internal structure for overlapping clusters and deterministic constraints. *JAIR* 27:1–23, 2006.

[13] Henrion. Propagating uncertainty in bayesian networks by probabilistic logic sampling. *UAI* 1988.

[14] Jensen, Lauritzen, and Olesen. Bayesian updating in causal probabilistic networks by local computations. *Comp. Stat. Quaterly* 4:269–282, 1990.

[15] Jojic, Jojic, Meek, Geiger, Siepel, Haussler, and Heckerman. Efficient approximations for learning phylogenetic hmm models from data. *ISMB* 2004.

[16] Jordan, Ghahramani, Jaakkola, and Saul. An introduction to variational methods for graphical models. *Machine Learning* 37(2):183–233, 1999.

[17] Mateescu, Dechter, and Kask. Partition-based anytime approximation for belief updating. 2001.

[18] Boyd and Vandenberghe. *Convex Optimization*. Cambridge University Press, 2004.

[19] Pearl. *Probabilistic Reasoning in Intelligent Systems*. Morgan Kaufmann, 1988.

[20] Shachter, D'Ambrosio, and Del Favero. Symbolic probabilistic inference in belief networks. *AAAI* 1990.

[21] Shachter and Peot. Simulation approaches to general probabilistic inference on belief networks.*UAI* 1989.

[22] Siepel and Haussler. Combining phylogenetic and HMMs in biosequence analysis. *RECOMB* 2003.

[23] Wainwright, Jaakkola, and Willsky. A new class of upper bounds on the log partition function. *IEEE Trans. Info. Theory* 51(7):2313–2335, 2005.

[24] Weiss. Belief propagation and revision in networks with loops. Technical Report AIM-1616, 1997.

[25] Wexler and Geiger. Importance sampling via variational optimization. *UAI* 2007.

[26] Wexler and Geiger. Variational upper bounds for probabilistic phylogenetic models. *RECOMB* 2007.

[27] Wexler and Meek. Inference for multiplicative models. *UAI* 2008.

[28] Xing, Jordan, and Russell. Graph partition strategies for generalized mean field inference. *UAI* 2004.
